# Extracting State Transition Dynamics from Multiple Spike Trains with Correlated Poisson HMM

**Kentaro Katahira[1,2], Jun Nishikawa[2], Kazuo Okanoya[2] and Masato Okada[1,2]**
[1]Graduate School of Frontier Sciences The University of Tokyo
Kashiwa, Chiba 277-8561, Japan
[2]RIKEN Brain Science Institute
Wako, Saitama 351-0198, Japan
`katahira@mns.k.u-tokyo.ac.jp`

## Abstract

Neural activity is non-stationary and varies across time. Hidden Markov Models (HMMs) have been used to track the state transition among quasi-stationary discrete neural states. Within this context, independent Poisson models have been used for the output distribution of HMMs; hence, the model is incapable of tracking the change in correlation without modulating the firing rate. To achieve this, we applied a multivariate Poisson distribution with correlation terms for the output distribution of HMMs. We formulated a Variational Bayes (VB) inference for the model. The VB could automatically determine the appropriate number of hidden states and correlation types while avoiding the overlearning problem. We developed an efficient algorithm for computing posteriors using the recursive relationship of a multivariate Poisson distribution. We demonstrated the performance of our method on synthetic data and a real spike train recorded from a songbird.

## 1   Introduction

Neural activities are highly non-stationary and vary from time to time according to stimuli and internal state changes. Hidden Markov Models (HMMs) have been used for segmenting spike trains into quasi-stationary states, in which the spike train is regarded as stationary, hence the statistics (e.g., cross-correlation and inter-spike interval) can be calculated [1, 2, 3]. We can also calculate these statistics by using time-binned count data (e.g., the Peri-Stimulus Time Histogram or PSTH). However, we need a large trial set to obtain good estimates for all bins, which can be problematic in neurophysiological experiments. HMMs enlarge the effective amount of data for estimating the statistics. Moreover, the PSTH approach cannot be applied to cases where we cannot align spike data to stimuli or the behaviors of animals. HMMs are suitable for such situations.

Previous studies using HMMs have assumed that all neural activities were independent of one another given the hidden states; hence, the models could not discriminate states whose firing rates were almost the same but whose correlations among neurons were different. However, there has been reports that shows the correlation between neurons changes within a fraction of a second without modulating the firing rate (e.g., [4]). We developed a method that enabled us to segment spike trains based on differences in neuronal correlation as well as the firing rate. Treating neuronal correlations (including higher-order, and not only pairwise correlations) among multiple spike trains has been one of the central challenges in computational neuroscience. There have been approaches to calculating correlations by binarizing spike trains with small bin sizes [5, 6]. These approaches are limited to treating correlations of short bin length that includes at most one spike. Here, we introduce a multivariate Poisson distribution with a higher-order correlation structure (simply abbreviated as a correlated Poisson distribution) as the output distribution for HMMs. The correlated Poisson distribution can incorporate correlation at arbitrary time intervals.

To construct optimal model from limited neurophysiological data, it is crucial to select a model that has appropriate complexity, and avoid over-fitting. In our model, model complexity corresponds to the number of hidden states and types of correlations (we have a choice as to whether to include pairwise correlation, third-order correlation, or higher order correlation). The maximum likelihood approach adopted in previous studies [1, 7, 8] cannot be used for this purpose since the likelihood criterion simply increases as the number of model parameters increases. A number of model-selection criteria used with the maximum likelihood approach, i.e., Akaike's information criteria (AIC), minimum description length (MDL), and Bayesian information criteria (BIC) are based on the asymptotic assumption that only holds when a large number of data is obtained. Furthermore, asymptotic normality, which is assumed in these criteria, does not hold in non-identifiable models including HMMs [9].

In this study, we applied the variational Bayes (VB) method [10, 11] to HMMs whose output distribution is a correlated Poisson distribution. VB is one of the approximations of the Bayes method and can avoid over-fitting even when the sample size is small. An optimal model structure can be determined based on tractable variational free energy, which is the upper bound of the negative marginal log-likelihood. Since the variational free energy does not need the asymptotic assumption, VB works well even when the sample size is small in practice [12]. The computation of posteriors for a correlated Poisson distribution imposes serious computational burdens. We developed an efficient algorithm to calculate these by using the recurrence relationship of a multivariate Poisson distribution [13]. To the best of our knowledge, this is the first report that has introduced VB method for a correlated Poisson distribution. Although Markov chain Monte Carlo (MCMC) methods has been applied to inferring posteriors for a correlated Poisson distribution [14], MCMC schemes are computationally demanding.

We demonstrate the performance of the method on multiple spike data both on a synthesized spike train and real spike data recorded from the forebrain nucleus for the vocal control (HVC) of an anesthetized songbird.

## 2   Method

### 2.1   HMM with multivariate Poisson distribution

Suppose that we obtain spike trains of $C$ neurons by using simultaneous recordings. As pre-processing, we first discretize the spike trains with a non-overlapping window whose length is $\Delta$ to obtain spike-count data. The number of spikes of neurons $c$ in the $t$th window of the $n$th trial is denoted by $x_c^{n,t}$. The spike-count data are summarized as $X^{n,t} = \{x_c^{n,t}\}_{c=1}^C$ and $X = \{X^{n,t}\}_{n=1,t=1}^{N,T}$. Let us assume spike-count data-set $X$ is produced by a $K$-valued discrete hidden state, $Y = \{y^{n,t}\}_{n=1,t=1}^{N,T}$, and the sequences of hidden states are generated by a first-order Markov process whose state transition matrix is $\mathbf{a} = \{a_{ij}\}_{i=1,j=1}^{K,K}$: $a_{ij} = p(y^{n,t} = j|y^{n,t-1} = i), \forall_{n,t}$ and the initial state probability is $\pi = \{\pi_i\} : \pi_i = p(y^{n,1} = i), \forall_n$, where $\sum_{i=1}^K \pi_i = 1, \sum_{j=1}^K a_{ij} = 1$, $a_{ij} \geq 0, \forall_{i,j}$. Hidden states $y_t$ are represented by a binary variable $y_k^{n,t}$ such that if the hidden state at the $t$th window of the $n$th trial is $k$, then $y_k^{n,t} = 1$; otherwise 0. At state $k$, the spike count is assumed to be generated according to $p(x_c^{n,t}|\lambda_k)$, whose specific form is given in the following.

Next, we introduce the correlated Poisson distribution. For brevity, we have omitted the superscript, $n, t$, for the moment. As an example, let us first consider cases of the trivariate Poisson model ($C = 3$) with second- and third-order correlations. We will introduce an auxiliary hidden variable, $s_l, l \in \Phi \equiv \{1, 2, 3, 12, 13, 23, 123\}$ which satisfies

$$\begin{aligned} x_1 &= s_1 + s_{12} + s_{13} + s_{123}, \\ x_2 &= s_2 + s_{12} + s_{23} + s_{123}, \\ x_3 &= s_3 + s_{13} + s_{23} + s_{123}. \end{aligned}$$

Each $s_l$ obeys $\mathcal{P}(s_l|\lambda_l)$, where $\mathcal{P}(x|\lambda)$ denotes a univariate Poisson distribution: $\mathcal{P}(x|\lambda) = \frac{\lambda^x}{x!}e^{-\lambda}$. Due to the reproducing properties of the Poisson distribution, each $x_i$ also marginally follows a Poisson distribution with parameter $\lambda_i + \lambda_{ij} + \lambda_{ik} + \lambda_{ijk}, i, j, k \in \{1, 2, 3\}, i \neq j \neq k$. The mean vector of this distribution is $(\lambda_1 + \lambda_{12} + \lambda_{13} + \lambda_{123}, \lambda_2 + \lambda_{12} + \lambda_{23} + \lambda_{123}, \lambda_3 + \lambda_{13} + \lambda_{23} + \lambda_{123})^T$

($T$ denotes the transposition) and its variance-covariance matrix is given by

$$
\begin{pmatrix}
\lambda_1 + \lambda_{12} + \lambda_{13} + \lambda_{123} & \lambda_{12} + \lambda_{123} & \lambda_{13} + \lambda_{123} \\
\lambda_{12} + \lambda_{123} & \lambda_2 + \lambda_{12} + \lambda_{23} + \lambda_{123} & \lambda_{23} + \lambda_{123} \\
\lambda_{13} + \lambda_{123} & \lambda_{23} + \lambda_{123} & \lambda_3 + \lambda_{13} + \lambda_{23} + \lambda_{123}
\end{pmatrix}.
$$

The general definition of the multivariate Poisson distribution is given using the vector, $S = (s_1, s_2, ..., s_L)^T$, and $C \times L$ matrix $B = [B_1, B_2, ..., B_J]$, $C \le L$ with 0 and 1 elements, where $B_j, j = 1, ..., J$ is a sub-matrix of dimensions $C \times {_C}\mathrm{C}_j$, where ${_C}\mathrm{C}_j$ is the number of combinations of choosing $j$ from $C$ elements. Vector $\mathbf{x} = (x_1, x_2, ..., x_C)^T$ defined as $\mathbf{x} = BS$ follows a multivariate Poisson distribution. In the above trivariate example, $S = (s_1, s_2, s_3, s_{12}, s_{13}, s_{23}, s_{123})^T$ and $B = [B_1, B_2, B_3]$, where

$$
B_1 = \begin{pmatrix} 1 & 0 & 0 \\ 0 & 1 & 0 \\ 0 & 0 & 1 \end{pmatrix}, \quad B_2 = \begin{pmatrix} 1 & 1 & 0 \\ 1 & 0 & 1 \\ 0 & 1 & 1 \end{pmatrix}, \quad B_3 = \begin{pmatrix} 1 \\ 1 \\ 1 \end{pmatrix}. \tag{1}
$$

We can also consider only the second-order correlation model by setting $B = [B_1, B_2]$ and $S = (s_1, s_2, s_3, s_{12}, s_{13}, s_{23})^T$, or only the third-order correlation model by setting $B = [B_1, B_3]$ and $S = (s_1, s_2, s_3, s_{123})^T$. The probability mass function of $\mathbf{x}$ is given by

$$
p(\mathbf{x}|\lambda_k) = \sum_{S \in G(\mathbf{x})} \prod_{l \in \Phi} \mathcal{P}(s_l|\lambda_{k,l}), \tag{2}
$$

where $G(\mathbf{x})$ denotes the set of $S$ such that $\mathbf{x} = BS$. The calculation of this probability can be computationally expensive, since summations over possible $S$ might be exhaustive, especially when there is a large number of spikes per window. However, the computational burden can be alleviated by using recurrence relations for a multivariate Poisson distribution [13]. For further details on computation, see the Appendix. We call the HMM with this output distribution the Correlated Poisson HMM (CP-HMM). When we assume that the spike counts for all neurons are independent, (i.e., $B = B_1$, $S = (s_1, s_2, s_3)^T$), the output distribution is reduced to

$$
p(\mathbf{x}|\lambda_k) = \prod_{c=1}^{C} \mathcal{P}(x_c|\lambda_{k,c}). \tag{3}
$$

We call the HMM with this distribution the independent Poisson HMM (IP-HMM). IP-HMM is a special case of CP-HMM. The complete log-likelihood for CP-HMM is

$$
\log p(X, Y, S|\theta) = \sum_{n=1}^{N} \left[ \sum_{k=1}^{K} y_k^{n,1} \log \pi_k + \sum_{t=2}^{T} \sum_{k=1}^{K} \sum_{k'=1}^{K} y_k^{n,t-1} y_{k'}^{n,t} \log a_{kk'} \right.
$$
$$
\left. + \sum_{t=1}^{T} \sum_{k=1}^{K} y_k^{n,t} \log \left\{ \mathbf{1}_{S^{n,t}}[G(X^{n,t})] \prod_{l \in \Phi} \mathcal{P}(s_l^{n,t}|\lambda_{k,l}) \right\} \right], \tag{4}
$$

where $\theta = (\pi, \mathbf{a}, \lambda)$ and $\mathbf{1}_A[x]$ is an indicator function, which equals 1 if $A \in x$ and 0 otherwise.

## 2.2 Variational Bayes

Here, we derive VB for CP-HMMs. We use conjugate prior distributions for all parameters of CP-HMMs, which enabled the posterior distribution to have the same form as the prior distribution. The prior distribution for initial probability distribution $\pi$ and state transition matrix $\mathbf{a}$ is the Dirichlet distribution:

$$
\varphi(\pi) = \mathcal{D}(\{\pi_k\}_{k=1}^{K}|\{u_k^{(\pi)}\}_{k=1}^{K}), \quad \varphi(\mathbf{a}) = \prod_{i=1}^{K} \mathcal{D}(\{a_{ik}\}_{k=1}^{K}|\{u_k^{(A)}\}_{k=1}^{K}). \tag{5}
$$

where $\mathcal{D}(\cdot)$ is defined as $\mathcal{D}(\{a_k\}_{k=1}^{K}|\{u_k\}_{k=1}^{K}) = \frac{\Gamma(\sum_{k=1}^{K} u_k)}{\prod_{k=1}^{K} \Gamma(u_k)} \prod_{k=1}^{k} a_k^{u_k-1}$. The conjugate prior for the parameter of the Poisson mean, $\lambda = \{\lambda_{k,l}\}_{k=1,l=1}^{K}$, of each auxiliary hidden variable, $\{s_l\}_{l \in \Phi}$, is

$$
\varphi(\lambda) = \prod_{k=1}^{K} \prod_{l \in \Phi} \mathcal{G}(\lambda_{k,l}|\kappa_0, \xi_0), \tag{6}
$$

where $\mathcal{G}(\cdot)$ denotes the Gamma distribution defined as $\mathcal{G}(\lambda|\kappa,\xi) = \frac{\xi^\kappa}{\Gamma(\kappa)}\lambda^{\kappa-1}e^{-\lambda\xi}$. In the experiments we discuss in the following, we set the hyperparameters as $u_j^{(\pi)} = u_j^{(A)} = 0.1, \forall j$, and $\kappa_0 = 0.1, \xi_0 = 0.1$.

The Bayesian method calculates $p(\theta, Z|X, M)$, which is a posterior of unknown parameters and hidden variable set $Z = (Y, S)$ given the data and model structure, $M$ (in our case, this indicates the number of hidden states, and correlation structure). However, the calculation of the posterior involves a difficult integral. The VB approach approximates the true posterior, $p(\theta, Z|X, M)$, by factored test distribution $r(\theta)Q(Z)$. To make the test distribution closer to the true posterior, we need to minimize Kullback-Leibler (KL) divergence from $r(\theta)Q(Z)$ to $p(\theta, Z|X, M)$:

$$\mathrm{KL}(r(\theta)Q(Z)||p(\theta, Z|X, M)) \equiv \left\langle \log \frac{r(\theta)Q(Z)}{p(Z, \theta|X, M)} \right\rangle_{r(\theta)Q(Z)}$$
$$= \log p(X|M) - \langle \log p(X, Z, \theta|M)\rangle_{r(\theta)Q(Z)} - \mathcal{H}_r(\theta) - \mathcal{H}_Q(Z), \qquad (7)$$

where $\langle \cdot \rangle_{p(x)}$ denotes the expectation over $p(x)$ and $\mathcal{H}_p(x) = \langle -\log p(x)\rangle_{p(x)}$ is the entropy of the distribution, $p(x)$. Since the log marginal likelihood $\log p(X|M)$ is independent of $r(\theta)$ and $Q(Z)$, minimizing KL divergence is equivalent to minimizing variational free energy

$$\mathcal{F} \equiv -\langle \log p(X, Z, \theta|M)\rangle_{r(\theta)Q(Z)} - \mathcal{H}_r(\theta) - \mathcal{H}_Q(Z). \qquad (8)$$

VB alternatively minimizes $\mathcal{F}$ with respect to $Q(Z)$ and $r(\theta)$. This minimization with respect to $Q(Z)$ is called the VB-E step, and the VB-M step for $r(\theta)$.

**VB-E step**   By using the Lagrange multiplier method, the VB-E step is derived as

$$Q(Z) \quad = \quad \frac{1}{C_Q} \exp\langle \log p(X, Z|\theta)\rangle_{r(\theta)},$$

where $C_Q$ is a normalization constant. More specifically, the following quantities are calculated:

$$\langle y_k^{n,t} \rangle_{Q(Z)} \quad = \quad \frac{\tilde{p}(y_k^{n,t} = 1|X^{n,1:t})\tilde{p}(X^{n,t+1:T}|y_k^{n,t} = 1)}{\sum_{i=1}^K \tilde{p}(y_i^{n,t} = 1|X^{n,1:t})\tilde{p}(X^{n,t+1:T}|y_i^{n,t} = 1)}$$

$$\langle y_k^{n,t-1} y_{k'}^{n,t} \rangle_{Q(Z)} \quad = \quad \frac{\tilde{p}(y_k^{n,t-1} = 1|X^{n,1:t-1})\tilde{a}_{kk'}\tilde{p}(X^{n,t}|\lambda_k')\tilde{p}(X^{n,t+1:T}|y_{k'}^{n,t} = 1)}{\sum_{i=1}^K \sum_{j=1}^K \tilde{p}(y_i^{n,t-1} = 1|X^{n,1:t-1})\tilde{a}_{ij}\tilde{p}(X^{n,t}|\lambda_j)\tilde{p}(X^{n,t+1:T}|y_j^{n,t} = 1)}$$

These quantities are obtained by the forward-backward algorithm [11]. The subnormarized quantity $\tilde{a}_{ij}$ is defined as $\tilde{a}_{ij} = \exp(\langle \log a_{ij}\rangle_{r(\mathbf{a})})$ and $\tilde{p}(X^{n,t}|\lambda_k)$ is

$$\tilde{p}(X^{n,t}|\lambda_k) = \sum_{S_k^{n,t} \in G(X^{n,t})} \prod_{l \in \Phi} \tilde{\mathcal{P}}(s_{k,l}^{n,t}|\lambda_{k,l}), \qquad (9)$$

where $\tilde{\mathcal{P}}(s_l|\lambda_{k,l})$ is a sub-normalized distribution:

$$\tilde{\mathcal{P}}(s_l|\lambda_{k,l}) = \exp\left\{s_l \log \tilde{\lambda}_{k,l} - \log(s_l!) - \bar{\lambda}_{k,l}\right\}, \qquad (10)$$

where

$$\tilde{\lambda}_{k,l} = \exp\left\{\langle \log \lambda_{k,l}\rangle_{r(\lambda_k)}\right\}, \quad \bar{\lambda}_{k,l} = \langle \lambda_{k,l}\rangle_{r(\lambda_k)}.$$

These quantities can be calculated by using the recurrence relations of the multivariate Poisson distribution (See the Appendix). The calculation of the posterior for $S$ is given as:

$$\langle s_{k,l}^{n,t} \rangle_{Q(Z)} = \langle y_k^{n,t} \rangle_{Q(Z)} \frac{\sum_{S_k^{n,t} \in G(X^{n,t})} s_{k,l}^{n,t} \prod_{l \in \Phi} \tilde{\mathcal{P}}(s_{k,l}^{n,t}|\lambda_{k,l})}{\sum_{S_k^{n,t} \in G(X^{n,t})} \prod_{l \in \Phi} \tilde{\mathcal{P}}(s_{k,l}^{n,t}|\lambda_{k,l})}. \qquad (11)$$

This is also calculated by using the recurrence relations of the multivariate Poisson distribution.

**VB-M step**   By again using the Lagrange multiplier method, the VB-M step is derived as

$$r(\theta) \quad = \quad \frac{1}{C_r}\varphi(\theta)\exp\langle \log p(X, Z|\theta)\rangle_{Q(Z)},$$

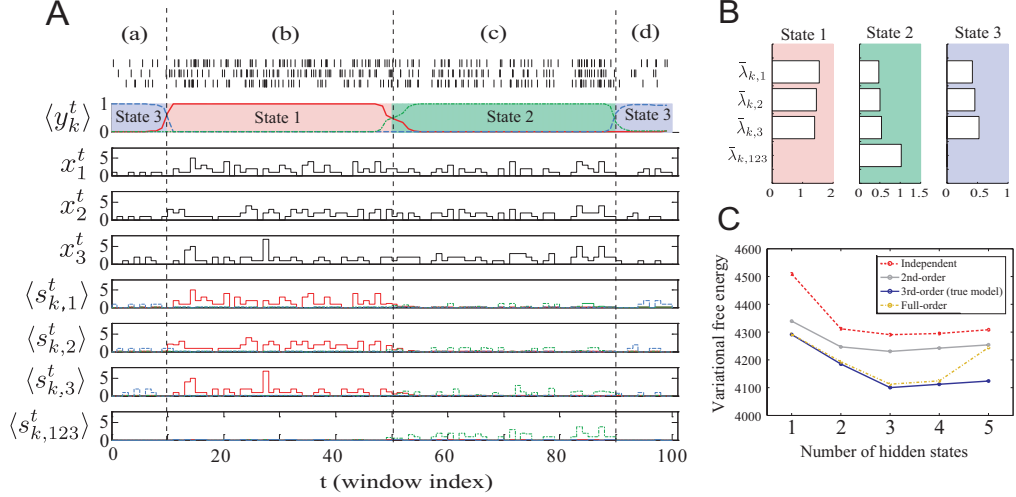

Figure 1: Typical examples of estimation results for correlated Poisson-HMM with third-order correlation applied to simulated spike train. **A**: From top, 1) spike train of three neurons, 2) the probability of state $k$ staying at window $t$ denoted by $\langle y_k^t \rangle_{Q(Z)}$, 3) spike count data $x_i^t$, and 4) posterior mean for hidden variables $s_{k,l}^t$. **B**: Posterior mean for Poisson mean $\lambda_{k,l}$ for all states. **C**: Variational free energy calculated for all models.

where $C_r$ is a normalization constant. More specifically, $r(\theta) = r(\pi)r(\mathbf{a})r(\lambda)$, and

$$r(\pi) = \mathcal{D}(\{\pi_k\}_{k=1}^K | \{w_k^\pi\}_{k=1}^K), \quad r(\mathbf{a}) = \prod_{i=1}^K \mathcal{D}(\{a_{ik}\}_{k=1}^K | \{w_{ik}^a\}_{k=1}^K),$$

$$r(\lambda) = \prod_{k=1}^K \prod_{l \in \Phi} \mathcal{G}(\lambda_{k,l} | w_{k,l}^\kappa, w_k^\xi),$$

where

$$w_{k,l}^\kappa = \kappa_0 + \sum_{n=1}^N \sum_{t=1}^T \langle s_{k,l}^{n,t} \rangle_{Q(Z)}, \quad w_k^\xi = \xi_0 + \sum_{n=1}^N \sum_{t=1}^T \langle y_k^{n,t} \rangle_{Q(Z)},$$

$$w_j^\pi = u_j^{(\pi)} + \sum_{n=1}^N \langle y_j^{n,1} \rangle_{Q(Z)}, \quad w_{ij}^a = u_j^{(a)} + \sum_{n=1}^N \sum_{t=2}^T \langle y_i^{n,t-1} y_j^{n,t} \rangle_{Q(Z)}.$$

The VB computes the VB-E and VB-M steps alternatively until the variational free energy converges to a local minimum. In the experiment we discuss in the following, we started the algorithm from 10 different initializations to avoid a poor local minimum solution.

## 3 Demonstration on synthetic spike train

By using the synthetic spike train of three neurons, let us first demonstrate how to apply our method to a spike train. In the case of three neurons, we have four choices for the correlation types that have (1) no correlation term, (2) only a second-order correlation term, (3) only a third-order correlation term, and (4) both of these. After this, we will call them IP-HMM, 2CP-HMM, 3CP-HMM, and full-CP-HMM. We generated spike trains by using a multivariate Poisson distribution with only a third-order correlation whose Poisson mean depends on periods as: (a) $\lambda_1 = \lambda_2 = \lambda_3 = 0.5$, $\lambda_{123} = 0.0$ for $t \in [1, 10]$, (b) $\lambda_1 = \lambda_2 = \lambda_3 = 1.5$, $\lambda_{123} = 0.0$ for $t \in [11, 50]$, (c) $\lambda_1 = \lambda_2 = \lambda_3 = 0.5$, $\lambda_{123} = 1.0$ for $t \in [51, 90]$, and (d) $\lambda_1 = \lambda_2 = \lambda_3 = 0.5$, and $\lambda_{123} = 0.0$ for $t \in [91, 100]$. The periods (b) and (c) have the same mean firing rate (the mean spike count in one window is $\lambda_i + \lambda_{123} = 1.5$, $i \in \{1, 2, 3\}$), but they only differ in the third-order correlation. Therefore, classical Poisson-HMMs that employ an independent Poisson assumption [1, 2, 7] are not able to segment them into distinct states. Figure 1A shows that our method was able to do so. We generated

Table 1: Results of model selection for spike trains from HVC

| Stimulus | $K$ | Correlation Structure |
|---|---|---|
| BOS | 4 | Independent |
| REV | 4 | 3rd-order |
| Silent | 3 | Independent |

Table 2: Results of model selection with time stationary assumption ($K = 1$)

| Stimulus | Correlation Structure |
|---|---|
| BOS | 2nd order |
| REV | 2nd order |
| Silent | Full order |

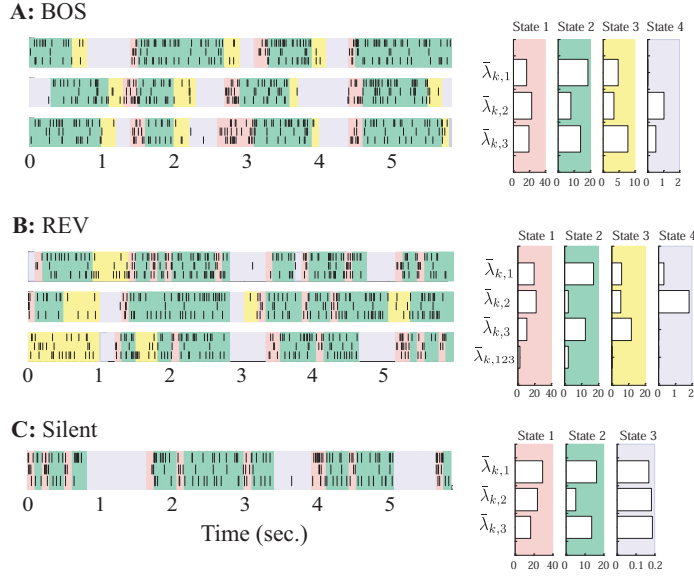

Figure 2: Typical examples of estimates of VB for spike train from HVC with (**A**) bird's own song, (**B**) its reversed song, and (**C**) no stimuli presented. Selected model based on variational free energy was used for each condition (see Table 1). Each row corresponds to different trials. Background color indicates most probable state at each time window. Right panels indicate posterior mean $\bar{\lambda}_{k,l}$ for all states.

spike trains for 10 trials, but only one trial is shown. The periods (b) and (c) are segmented into states 1 and 2, whose Poisson means are different (Fig. 1B). The bottom four lines in Fig. 1A plot the posterior mean for $\{s_{k,l}^t\}_{l \in \Phi}$ (Here, we omitted the index of trial $n$). These plots separately visualize the contribution of the independent factor and correlation factor on spike counts $x_c^t, c \in \{1, 2, 3\}$. The spike counts in period (b) can be viewed as independent firing. Even if the spikes are in the same window, this can be regarded as just a coincidence predicted by the assumption of independent firing. In contrast, the spike counts in period (c) can be regarded as having been contributed by common factor $s_{2,123}^t$, as well as independent factors $s_{2,i}^t, i \in \{1, 2, 3\}$. Here, we used a 3CP-HMM having three hidden states. Because periods (a) and (d) have identical statistics, it is clear that the model with three states ($K = 3$) is sufficient for modeling this spike train. Then, can we select this model from the data? Figure 1C shows the variational free energy, $\mathcal{F}$. The 3CP-HMM with three hidden states yields the lowest $\mathcal{F}$, implying that it is optimal. The 3CP-HMMs with fewer hidden states, IP-HMMs, or 2CP-HMMs cannot represent the statistical structure of the data, and hence yield higher $\mathcal{F}$. The 3CP-HMMs with more hidden states ($K > 3$) or full-CP-HMMs ($K \geq 3$) can include an optimal model, but by being penalized by a *Bayesian Occam's razor*, yield higher $\mathcal{F}$. Thus, we can select the optimal model based on $\mathcal{F}$, at least in this example.

## 4  Application to spike trains from HVC in songbird

We applied our method to data collected from the nucleus HVC of songbird. HVC is an important nucleus that integrates auditory information and motor information of song sequences [15]. We obtained spike trains of three single units by using a silicon probe from one anesthetized Bengalese finch. The bird's own song (BOS) and reversed song (REV) were presented 50 times for each

Table 3: Log-likelihood on test data (REV).

| Method | Log-likelihood (mean $\pm$ s.d.) |
|---|---|
| Independent & stationary assumption ($K = 1$) | -255.691 ($\pm$ 2.074) |
| Stationary assumption ($K = 1$, correlation type is selected) | -247.640 ($\pm$ 1.659) |
| Independent assumption (IP-HMM) ($K$ is selected) | -230.353 ($\pm$ 0.958) |
| CP-HMM (all selected) | **-229.143 ($\pm$ 1.242)** |
| full-CP-HMM ($K$ is selected) | -230.272 ($\pm$ 1.244) |

stimulus during recording. Spontaneous activities (Silent) were recorded so that we could obtain the same amount of data as the stimulus-presented data. More details on the recordings are described elsewhere [16].

We modeled spike trains for all stimuli using IP-HMMs and CP-HMMs by varying the number of states $K$ and various correlation structure. We then selected the model that yielded the lowest free energy. We used window length $\Delta = 100$ (ms). The selected models are summarized in Table 1. Figure 2 shows a typical example of spike trains and the segmentation results for the selected models. The CP-HMMs were only selected for spike trains when REV was presented. If we assume that the spike statistics did not change over the trials (in our case, this corresponds to the model with only one hidden state, $K = 1$), CP-HMMs were selected under all experimental conditions. These results reflect the fact that neurons in anesthetized animals simultaneously transit between high-firing and low-firing states [17], which can be captured by a Poisson distribution with correlation terms. Time-stationary assumptions have often been employed to obtain a sufficient sample size for estimating correlation (e.g., [6]). Our results suggest that we should be careful when interpreting such results; even when the spike trains seem to have a correlation, if we take state transition into account, spike trains may be better captured by using an independent Poisson model.

We measured predictive performance on test data to verify how well our model capture the statistical properties of the spike train. Here, we used spike trains for REV where 3CP-HMM was selected. We first divided spike trains into 20 training and 20 test trials. In the training phase, we constructed models using the model selection based on the variational free energy with four restrictions: (1) an independent & stationary assumption ($K = 1$), (2) a stationary assumption ($K = 1$, correlation type was selected), (3) IP-HMM ($K$ was selected), (4) CP-HMM (no restrictions), and (5) the full-CP-HMM ($K$ was selected). In the prediction phase, we calculated the log-likelihood on test data under the posterior mean $\langle\theta\rangle_{r(\theta)}$ of selected models. The results are summarized in Table 3. We took averaged over different choices of training set and prediction sets. We can see that the log-likelihood on the test data improved by taking both the state transition and correlation structure into consideration. These results imply that CP-HMMs can characterize the spike train better than classical Poisson-HMMs. The full-CP-HMM include 2nd-order CP-HMM, but shows lower predictive performance than the model in which correlation type were selected. This is likely due to over-fitting to the training data. The VB approach selected the model with tappropriate complexity avoiding over-fitting.

## 5 Discussion

We constructed HMMs whose output is a correlated multivariate Poisson distribution for extracting state-transition dynamics from multiple spike trains. We applied the VB method for inferring the posterior over the parameter and hidden variables of the models. We have seen that VB can be used to select an appropriate model (the number of hidden states and correlation structure), which gives a better prediction. Our method incorporated the correlated Poisson distribution for treating pairwise and higher-order correlations. There have been approaches that have calculated correlations by binarizing spike data with log-linear [5] or maximum-entropy models [6]. These approaches are limited to treating correlations in short bin lengths, which include at most one spike. In contrast, our approach can incorporate correlations in an arbitrary time window from exact synchronization to firing-rate correlations on a modest time scale. The major disadvantages of our model are that it is incapable of negative correlations. It can be incorporated by employing a mixture of multivariate-Poisson distributions for the output distribution of HMMs. VB can easily be extended to such models.

## Appendix: Calculation of correlated Poisson distribution in VB-E step

The sub-normalized distribution (Eq.9) can be calculated by using the recurrence relation of multivariate Poisson distribution [13]. Let us consider the tri-variate ($C = 3$) with the second-order correlation case, where $B = [B_1, B_2]$. Here, the recursive scheme for the calculating Eq.9 is:

- If all elements of $X = (X_1, X_2, X_3)$ are non-zero, then

$$
\begin{aligned}
x_1 \tilde{\mathcal{P}}(X_1 = x_1, X_2 = x_2, X_3 = x_3 | \lambda) =& \tilde{\lambda}_1 \tilde{\mathcal{P}}(X_1 = x_1 - 1, X_2 = x_2, X_3 = x_3 | \lambda) \\
&+ \tilde{\lambda}_{12} \tilde{\mathcal{P}}(X_1 = x_1 - 1, X_2 = x_2 - 1, X_3 = x_3 | \lambda) \\
&+ \tilde{\lambda}_{13} \tilde{\mathcal{P}}(X_1 = x_1 - 1, X_2 = x_2, X_3 = x_3 - 1 | \lambda).
\end{aligned}
$$

- If at most one element of $X$ is non-zero, then

$$
\tilde{\mathcal{P}}(X_1 = x_1, X_2 = x_2, X_3 = x_3 | \lambda) = \exp\left\{ -\sum_{i<j} \tilde{\lambda}_{ij} \right\} \prod_{i=1}^{3} \tilde{\mathcal{P}}(X_i = x_i | \lambda_i), \ \ i,j \in 1,2,3.
$$

- If only one of the $x_i$'s (say, $x_k$) is zero, then

$$
\tilde{\mathcal{P}}(X_1 = x_1, X_2 = x_2, X_3 = x_3 | \lambda) = \exp\{-\tilde{\lambda}_{ik} - \tilde{\lambda}_{jk}\} \tilde{\mathcal{P}}(X_i = x_i, X_j = x_j | \lambda_i, \lambda_j, \lambda_{ij}).
$$

This recursive scheme can be generalized to more than three dimensions. We use the alternative definition of the multivariate Poisson random vector, $\mathbf{x}$ such that $\mathbf{x} = \sum_{l=1}^{k} \phi_l s_l$, where the vectors, $\phi_l$, denote a $l$th column of matrix $B$. Let us define vector $\lambda^* = (\tilde{\lambda}_1 \tilde{\mathcal{P}}(X = \mathbf{x} - \phi_1 | \lambda), ..., \tilde{\lambda}_k \tilde{\mathcal{P}}(X = \mathbf{x} - \phi_k | \lambda))^T$. Then, the recurrence relations are rewritten as

$$
\mathbf{x} \tilde{\mathcal{P}}(X = \mathbf{x} | \lambda) = B \lambda^*. \tag{12}
$$

By using the quantities obtained in this calculation, $\langle s_{k,l}^{n,t} \rangle_{Q(Z)}$ is calculated as

$$
\langle s_{k,l}^{n,t} \rangle_{Q(S)} = \langle y_k^{n,t} \rangle_{Q(Z)} \frac{\tilde{\lambda}_{k,l} \tilde{\mathcal{P}}(X^{n,t} - \phi_l | \lambda_k)}{\tilde{\mathcal{P}}(X^{n,t} | \lambda_k)}. \tag{13}
$$

## References

[1] M. Abeles, H. Bergman, I. Gat, I. Meilijson, E. Seidemann, N. Thishby, and E. Vaadia, *Proc Nat Acad Sci USA*, 92:8616-8620, 1995.

[2] I. Gat, N. Tishby, and M. Abeles, *Network: Computation in Neural Systems*, 8:297-22, 1997.

[3] L. M. Jones, A. Fontanini, B. F. Sadacca, P. Miller, and D. B. Katz, *Proc Nat Acad Sci USA* 104:18772-18777, 2007.

[4] E. Vaadia, I. Haalman, M. Abeles. H. Bergman, Y. Prut, H. Slovin, and A. Aertsen, *Nature*, 373:515-518, 1995.

[5] H. Nakahara, and S. Amari, *Neural Computation* 14:2269-2316, 2002.

[6] E. Schneidman, M. J. Berry, R. Segev and W. Bialek, *Nature* 440:1007-1012, 2006.

[7] G. Radons, J.D. Becker, B. Dülfer, and J Krüger, *Biological Cybernetics*, 71:359-73, 1994.

[8] M. Danoczy and R. Hahnloser, *Advances in NIPS*, 18, 2005.

[9] K. Yamazaki and S. Watanabe, *Neurocomputing* 69:62-84, 2005.

[10] H. Attias, *in Proc. of 15th Conference on Uncertainty in Artificial Intelligence*, 21-30, 1999.

[11] M. J. Beal, Variational Algorithms for Approximate Bayesian Inference, *Ph.D thesis*, University College London, 2003.

[12] S. Watanabe, Y. Minami, A. Nakamura, and N. Ueda, *Advances in NIPS*, 15, 2002.

[13] K. Kano and K. Kawamura, *Communications in Statistics*, 20:165-178, 1991.

[14] L. Meligkotsidou, *Statistics and Computing*, 17:93-107, 2007

[15] A.C. Yu and D. Margoliash, *Science*, 273:1871-1875, 1996.

[16] J. Nishikawa and K. Okanoya, *in preparation*.

[17] G. Uchida, M. Fukuda, and M. Tanifuji, *Physical Review E*, 73:031910, 2006

